# From Coexpression to Coregulation: An Approach to Inferring Transcriptional Regulation among Gene Classes from Large-Scale Expression Data

**Eric Mjolsness**
Jet Propulsion Laboratory
California Institute of Technology
Pasadena CA 91109-8099
*mjolsness@jpl.nasa.gov*

**Tobias Mann**
Jet Propulsion Laboratory
California Institute of Technology
Pasadena CA 91109-8099
*mann@aig.jpl.nasa.gov*

**Rebecca Castaño**
Jet Propulsion Laboratory
California Institute of Technology
Pasadena CA 91109-8099
*becky@aig.jpl.nasa.gov*

**Barbara Wold**
Division of Biology
California Institute of Technology
Pasadena CA 91125
*woldb@its.caltech.edu*

## Abstract

We provide preliminary evidence that existing algorithms for inferring small-scale gene regulation networks from gene expression data can be adapted to large-scale gene expression data coming from hybridization microarrays. The essential steps are (1) clustering many genes by their expression time-course data into a minimal set of clusters of co-expressed genes, (2) theoretically modeling the various conditions under which the time-courses are measured using a continious-time analog recurrent neural network for the cluster mean time-courses, (3) fitting such a regulatory model to the cluster mean time courses by simulated annealing with weight decay, and (4) analysing several such fits for commonalities in the circuit parameter sets including the connection matrices. This procedure can be used to assess the adequacy of existing and future gene expression time-course data sets for determining transcriptional regulatory relationships such as coregulation.

## 1 Introduction

In a cell, genes can be turned "on" or "off" to varying degrees by the protein products of other genes. When a gene is "on" it is transcribed to produce messenger RNA (mRNA) which can subsequently be translated into protein molecules. Some of these proteins are transcription factors which bind to DNA at specific sites and thereby affect which genes are transcribed and how often. This trancriptional

regulation feedback circuitry provides a fundamental mechanism for information processing in the cell. It governs differentiation into diverse cell types and many other basic biological processes.

Recently, several new technologies have been developed for measuring the "expression" of genes as mRNA or protein product. Improvements in conventional fluorescently labeled antibodies against proteins have been coupled with confocal microscopy and image processing to partially automate the simultaneous measurement of small numbers of proteins in large numbers of individual nuclei in the fruit fly *Drosophila melanogaster* [1]. In a complementary way, the mRNA levels of thousands of genes, each averaged over many cells, have been measured by hybridization arrays for various species including the budding yeast *Saccharomyces cerevisiae* [2].

The high-spatial-resolution protein antibody data has been quantitatively modeled by "gene regulation network" circuit models [3] which use continuous-time, analog, recurrent neural networks (Hopfield networks without an objective function) to model transcriptional regulation [4][5]. This approach requires some machine learning technique to infer the circuit parameters from the data, and a particular variant of simulated annealing has proven effective [6][7]. Methods in current biological use for analysing mRNA hybridization data do not infer regulatory relationships, but rather simply cluster together genes with similar patterns of expression across time and experimental conditions [8][9]. In this paper, we explore the extension of the gene circuit method to the mRNA hybridization data which has much lower spatial resolution but can currently assay a thousand times more genes than immunofluorescent image analysis.

The essential problem with using the gene circuit method, as employed for immunoflourescence data, on hybridization data is that the number of connection strength parameters grows between linearly and quadratically in the number of genes (depending on sparsity assumptions) . This requires more data on each gene, and even if that much data is available, simulated annealing for circuit inference does not seem to scale well with the number of unknown parameters. Some form of dimensionality reduction is called for. Fortunately dimensionality reduction is available in the present practice of clustering the large-scale time course expression data by genes, into gene clusters. In this way one can derive a small number of cluster-mean time courses for "aggregated genes", and then fit a gene regulation circuit to these cluster mean time courses. We will discuss details of how this analysis can be performed and then interpreted. A similar approach using somewhat different algorithms for clustering and circuit inference has been taken by Hertz [10].

In the following, we will first summarize the data models and algorithms used, and then report on preliminary experiments in applying those algorithms to gene expression data for 2467 yeast genes [9][11]. Finally we will discuss prospects for and limitations of the approach.

## 2 Data Models and Algorithms

The data model is as follows. We imagine that there is a small, hidden regulatory network of "aggregate genes" which regulate one another by the analog neural network dynamics [3]

$$\tau_i \frac{dv_i}{dt} = g\left(\sum_j T_{ij} v_j + h_i\right) - \lambda_i v_i$$

in which $v_i$ is the continuous-valued state variable for gene product $i$, $T_{ij}$ is the matrix of positive, zero, or negative connections by which one transcription factor can enhance or repress another, and $g()$ is a nonlinear monotonic sigmoidal activation function. When a particular matrix entry $T_{ij}$ is nonzero, there is a regulatory "connection" from gene product $j$ to gene $i$. The regulation is enhancing if $T$ is positive and repressing if it is negative. If $T_{ij}$ is zero there is no connection.

This network is run forwards from some initial condition and time-sampled to generate a wild-type time course for the aggregate genes. In addition, various other time courses can be generated under alternative experimental conditions by manipulating the parameters. For example an entire aggregate gene (corresponding to a cluster of real genes) could be removed from the circuit or otherwise modified to represent mutants. External input conditions could be modeled as modifications to $h$. Thus we get one or several time courses (trajectories) for the aggregate genes.

From such aggregate time courses, actual gene data is generated by addition of Gaussian-distributed noise to the logarithms of the concentration variables. Each time point in each cluster has its own scalar standard deviation parameter (and a mean arising from the circuit dynamics). Optionally, each gene's expression data may also be multiplied by a time-independent proportionality constant.

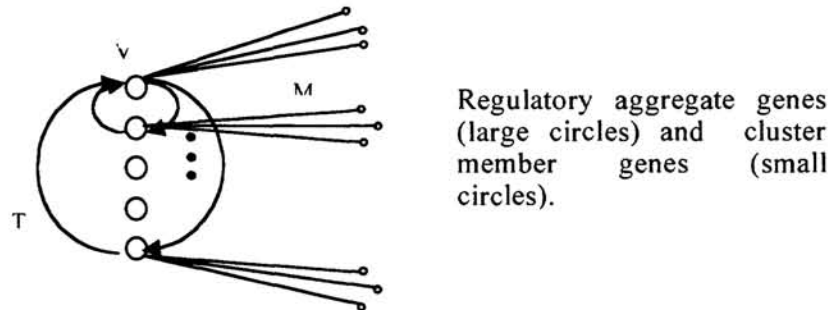

Regulatory aggregate genes (large circles) and cluster member genes (small circles).

Given this data generation model and suitable gene expression data, the problem is to infer gene cluster memberships and the circuit parameters for the aggregate genes' regulatory relationships. Then, we would like to use the inferred cluster memberships and regulatory circuitry to make testable biological predictions.

This data model departs from biological reality in many ways that could prove to be important, both for inference and for prediction. Except for the Gaussian noise model, each gene in a cluster is models as fully coregulated with every other one – they are influenced in the same ways by the same regulatory connection strengths. Also, the nonlinear circuit model must not only reflect transcriptional regulation, but all other regulatory circuitry affecting measured gene expression such as kinase-phosphatase networks.

Under this data model, one could formulate a joint Bayesian inference problem for the clustering and circuit inference aspects of fitting the data. But given the highly provisional nature of the model, we simply apply in sequence an existing mixture-of-Gaussians clustering algorithm to preprocess the data and reduce its dimensionality, and then an existing gene circuit inference algorithm. Presumably a joint optimization algorithm could be obtained by iterating these steps.

## 2.1   Clustering

A widely used clustering algorithm for mixure model estimation is Expectation-Maximization (EM)[12]. We use EM with a diagonal covariance in the Gaussian, so that for each feature vector component $a$ (a combination of experimental condition

and time point in a time course) and cluster $\alpha$ there is a standard deviation parameter $\sigma_{a\alpha}$. In preprocessing, each concentration data point is divided by its value at time zero and then a logarithm taken. The log ratios are clustered using EM. Optionally, each gene's entire feature vector may be normalized to unit length and the cluster centers likewise normalized during the iterative EM algorithm.

In order to choose the number of clusters, $k$, we use the cross-validation algorithm described by Smyth [13]. This involves computing the likelihood of each optimized fit on a test set and averaging over runs and over divisions of the data into training and test sets. Then, we can examine the likelihood as a function of $k$ in order to choose $k$. Normally one would pick $k$ so as to maximize cross-validated likelihood. However, in the present application we also want to reward small values of $k$ which lead to smaller circuits for the circuit inference phase of the algorithm. The choice of $k$ will be discussed in the next section.

## 2.2  Circuit Inference

We use the Lam-Delosme variant of simulated annealing (SA) to derive connection strengths $T$, time constants $\tau$, and decay rates $\lambda$, as in previous work using this gene circuit method [4][5]. We set $h$ to zero. The score function which SA optimizes is

$$S(T,\tau,\lambda) = A\sum_{it}\left(v_i(t;T,\tau,\lambda) - \hat{v}_i(t)\right)^2 + W\sum_{ij}T_{ij}^2$$
$$+ \exp[B(\sum_{ij}T_{ij}^2 + \sum_i\lambda_i^2 + \sum_i\tau_i^2)] - 1$$

The first term represents the fit to data $\hat{v}_i$. The second term is a standard weight decay term. The third term forces solutions to stay within a bounded region in weight space. We vary the weight decay coefficient $W$ in order to encourage relatively sparse connection matrix solutions.

# 3  Results

## 3.1  Data

We used the *Saccharomyces cerevisiae* data set of [9]. It includes three longer time courses representing different ways to synchronize the normal cell cycle [11], and five shorter time courses representing altered conditions. We used all eight time courses for clustering, but just 8 time points of one of the longer time courses (alpha factor synchronized cell cycle) for the circuit inference. It is likely that multiple long time courses under altered conditions will be required before strong biological predictions can be made from inferred regulatory circuit models.

## 3.2  Clustering

We found that the most likely number of classes as determined by cross validation was about 27, but that there is a broad plateau of high-likelihood cluster numbers from 15 to 35 (Figure 1). This is similar to our results with another gene expression data set for the nematode worm *Caenorhabditis elegans* supplied by Stuart Kim; these more extensive clustering experiments are summarized in Figure 2. Clustering experiments with synthetic data is used to understand these results. These experiments show that the cross-validated log likelihood curve can indicate the number of clusters present in the data, justifying the use of the curve for that

purpose. In more detail, synthetic data generated from 14 20-dimensional spherical Gaussian clusters were clustered using the EM/CV algorithm. The likelihoods showed a sharp peak at $k$=14 unlike Figures 1 or 2. In another experiment, 14 20-dimensional spherical Gaussian superclusters were used to generate second-level clusters (3 subclusters per supercluster), which in turn generated synthetic data points. This two-level hierarchical model was then clustered with the EM/CV method. The likelihood curves (not shown) were quite similar to Figures 1 and 2, with a higher-likelihood plateau from roughly 14 to 40.

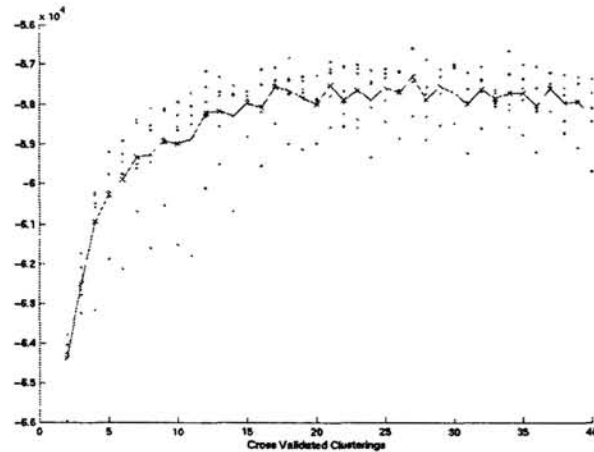

**Figure 1.** Cross-validated log-likelihood scores, displayed and averaged over 5 runs, for EM clustering of *S. cerevisiae* gene expression data [9]. Horizontal axis: $k$, the "requested" or maximal number of cluster centers in the fit. Some cluster centers go unmatched to data. Vertical axis: log likelihood score for the fit, scatterplotted and averaged. Likelihoods have not been integrated over any range of parameters for hypothesis testing. $k$ ranges from 2 to 40 in increments of 1. Solid line shows average likelihood value for each $k$.

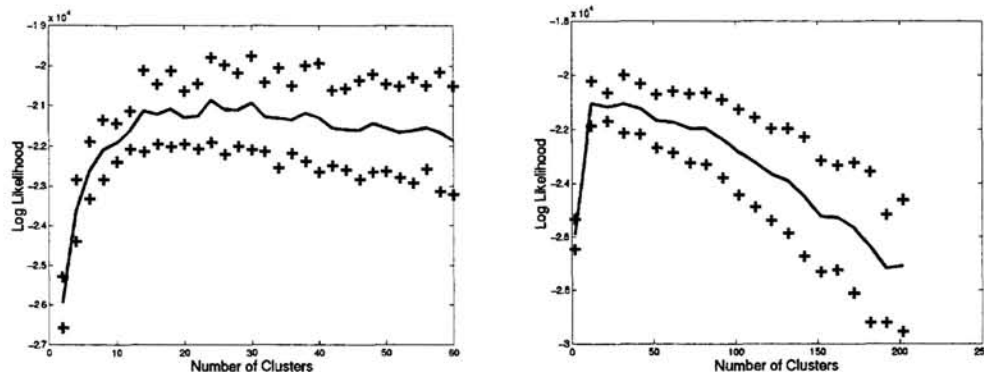

**Figure 2.** Cross-validated log-likelihood scores, averaged over 13 runs, for EM clustering of *C. elegans* gene expression data from S. Kim's lab. Horizontal axis: $k$, the "requested" or maximal number of cluster centers in the fit. Some cluster centers go unmatched to data. Vertical axis: log likelihood score for the fit, as an average over 13 runs plus or minus one standard deviation. (Left) Fine-scale plot, $k$ =2 to 60 in increments of 2. (Right) Coarse-scale plot, $k$=2 to 202 in increments of 10. Both plots show an extended plateau of relatively likely fits between roughly $k$ =14 and $k$ =40.

From Figures 1 and 2 and the synthetic data experiments mentioned above, we can guess at appropriate values for $k$ which take into account both the measured likelihood of clustering and the requirements for few parameters in circuit-fitting. For example choosing $k$=15 clusters would put us at the beginning of the plateau, losing very little cluster likelihood in return for reducing the aggregate genes circuit size from 27 to 15 players. The interpretation would be that there are about 15 superclusters in hierarchically clustered data, to which we will fit a 15-player

regulatory circuit. Much more aggressive would be to pick $k=7$ or 8 clusters, for a relatively significant drop in log-likelihood in return for a further substantial decrease in circuit size. An acceptable range of cluster numbers (and circuit sizes) would seem to be $k=8$ to 15.

## 3.3   Gene Circuit Inference

It proved possible to fit the $k=15$ time course using weight decay $W=1$ but without using hidden units. $W=0$ and $W=3$ gave less satisfactory results. Four of the 15 clusters are shown in Figure 3 for one good run ($W=1$). Scores for our first few (unselected) runs at the current parameter settings are shown in Table 1. Each run took between 24 and 48 hours on one processor of an Sun Ultrasparc 60 computer. Even with weight decay, it is possible that successful fits are really overfits with this particular data since there are about twice as many parameters as data points.

| Weight Decay $W$ | \<Score\> | \<Simulated Annealing Moves\>/$10^6$ | Number of runs |
|---|---|---|---|
| 0 | 1.536 +/- 0.134 | 2.803 +/- 0.437 | 3 |
| 1 | 0.787 +/- 0.394 | 2.782 +/- 0.200 | 10 |
| 3 | 1.438 +/- 0.037 | 2.880 +/- 0.090 | 4 |

**Table 1**. Score function parameters were A=1.0, B=0.01. Annealing runs statistics are reported when the temperature dropped below 0.0001. All the best scores and visually acceptable fits occurred in W=1 runs.

The average values of the data fit, weight decay, and penalty terms in the score function for W=1 were {0.378, 0.332, 0.0667} after slightly more annealing.

There were a few significant similarities between the connection matrices computed in the two lowest-scoring runs. The most salient feature in the lowest-scoring network was a set of direct feedback loops among its strongest connections: cluster 8 both excited and was inhibited by cluster 10, and cluster 10 excited and was inhibited by cluster 15. This feature was preserved in the second-best run. A systematic search for "concensus circuitry" shows convergence towards a unique connection matrix for the 8-point time series data used here, but more complete 16-time-point data gives multiple "clusters" of connection matrices. From parameter-counting one might expect that making robust and unique regulatory predictions will require the use of more trajectory data taken under substantially different conditions. Such data is expected to be forthcoming.

## 4   Discussion

We have illustrated a procedure for deriving regulatory models from large-scale gene expression data. As the data becomes more comprehensive in the number and nature of conditions under which comparable time courses are measured, this procedure can be used to determine when biological hypotheses about gene regulation can be robustly derived from the data.

### Acknowledgments

This work was supported in part by the Whittier Foundation, the Office of Naval Research under contract N00014-97-1-0422, and the NASA Advanced Concepts Program. Stuart Kim (Stanford University) provided the *C. elegans* gene expression array data. The GRN simulation and inference code is due in part to Charles Garrett and George Marnellos. The EM clustering code is due in part to Roberto Manduchi.

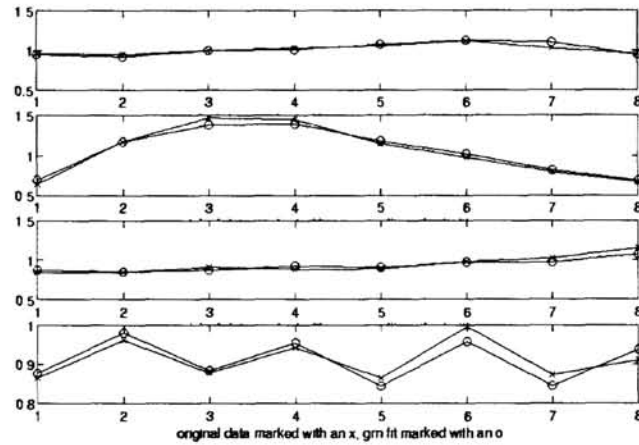

**Figure 3**. Four clusters (numbers 9-12) of a 15-cluster mixture of Gaussians model of 2467 genes each assayed over an eight-point time course; cluster means (shown as x) are fit to a gene regulation network model (shown as o).

## References

[1] D. Kosman, J. Reinitz, and D. H. Sharp, "Automated Assay of Gene Expression at Cellular Resolution" Pacific Symposium on Biocomputing '98. Eds. R. Altman, A. K. Dunker, L. Hunter, and T. E. Klein,, World Scientific 1998.

[2] J. L. DeRisi, V. R. Iyer, and P. O. Brown, "Exploring the Metabolic and Genetic Control of Gene Expreession on a Genomic Scale". Science 278, 680-686.

[3] A Connectionist Model of Development, E. Mjolsness, D. H. Sharp, and J. Reinitz, Journal of Theoretical Biology 152:429-453, 1991.

[4] J. Reinitz, E. Mjolsness, and D. H. Sharp, "Model for Cooperative Control of Positional Information in *Drosophila* by Bicoid and Maternal Hunchback". J. Experimental Zoology 271:47-56, 1995. Los Alamos National Laboratory Technical Report LAUR-92-2942 1992.

[5] J. Reinitz and D. H. Sharp, "Mechanism of *eve* Stripe Formation". Mechanisms of Development 49:133-158, 1995.

[6] [7] J. Lam and J. M. Delosme. "An Efficient Simulated Annealing Schedule: Derivation" and "... Implementation and Evaluation". Technical Reports 8816 and 8817, Yale University Electrical Engineering Department, New Haven CT 1988.

[8] X. Wen, S. Fuhrman, G. S. Michaels, D. B. Carr, S. Smith, J. L. Barker, and R. Somogyi, "Large-Scale Temporal Gene Expression Mapping of Central Nervous System Development", Proc. Natl. Acal. Sci. USA 95:334-339, January 1998.

[9] M. B. Eisen, P. T. Spellman, P. O. Brown, and D. Botstein, "Cluster Analysis and Display of Genome-Wide Expression Patterns", Proc. Natl. Acad. Scie. USA 95:14863-14868, December 1998.

[10] J. Hertz, lecture at Krogerrup Denmark computational biology summer school, July 1998.

[11] Spellman PT, Sherlock G, Zhang MQ, et al., "Comprehensive identification of cell cycle-regulated genes of the yeast Saccharomyces cerevisiae by microarray hybridization", Mol. Bio. Cell. 9: (12) 3273-3297 Dec 1998.

[12] Dempster, A. P., Laird, N. M. and Rubin, D. B. "Maximum likelihood from incomplete data via the EM algorithm," J. Royal Statistical Society, Series B, 39:1-38, 1977.

[13] P. Smyth, "Clustering using Monte Carlo Cross-Validation", Proceedings of the 2nd International Conference on Knowledge Discovery and Data Mining, AAAI Press, 1996.
